# A Recurrent Model of Orientation Maps with Simple and Complex Cells

**Paul Merolla and Kwabena Boahen**
Department of Bioengineering
University of Pennsylvania
Philadelphia, PA 19104
*{pmerolla,boahen} @seas.upenn.edu*

## Abstract

We describe a neuromorphic chip that utilizes transistor heterogeneity, introduced by the fabrication process, to generate orientation maps similar to those imaged *in vivo*. Our model consists of a recurrent network of excitatory and inhibitory cells in parallel with a push-pull stage. Similar to a previous model the recurrent network displays hotspots of activity that give rise to visual feature maps. Unlike previous work, however, the map for orientation does not depend on the sign of contrast. Instead, sign-independent cells driven by both ON and OFF channels anchor the map, while push-pull interactions give rise to sign-preserving cells. These two groups of orientation-selective cells are similar to complex and simple cells observed in V1.

## 1 Orientation Maps

Neurons in visual areas 1 and 2 (V1 and V2) are selectively tuned for a number of visual features, the most pronounced feature being orientation. Orientation preference of individual cells varies across the two-dimensional surface of the cortex in a stereotyped manner, as revealed by electrophysiology [1] and optical imaging studies [2]. The origin of these preferred orientation (PO) maps is debated, but experiments demonstrate that they exist in the absence of visual experience [3]. To the dismay of advocates of Hebbian learning, these results suggest that the initial appearance of PO maps rely on neural mechanisms oblivious to input correlations. Here, we propose a model that accounts for observed PO maps based on innate noise in neuron thresholds and synaptic currents. The network is implemented in silicon where heterogeneity is as ubiquitous as it is in biology.

## 2 Patterned Activity Model

Ernst et al. have previously described a 2D rate model that can account for the origin of visual maps [4]. Individual units in their network receive isotropic feedforward input from the geniculate and recurrent connections from neighboring

units in a Mexican hat profile, described by short-range excitation and long-range inhibition. If the recurrent connections are sufficiently strong, hotspots of activity (or 'bumps') form periodically across space. In a homogeneous network, these bumps of activity are equally stable at any position in the network and are free to wander.

Introducing random jitter to the Mexican hat connectivity profiles breaks the symmetry and reduces the number of stable states for the bumps. Subsequently, the bumps are pinned down at the locations that maximize their net local recurrent feedback. In this regime, moving gratings are able to shift the bumps away from their stability points such that the responses of the network resemble PO maps. Therefore, the recurrent network, given an ample amount of noise, can innately generate its own orientation specificity without the need for specific hardwired connections or visually driven learning rules.

## 2.1    Criticisms of the Bump model

We might posit that the brain uses a similar opportunistic model to derive and organize its feature maps – but the parallels between the primary visual cortex and the Ernst et al. bump model are unconvincing. For instance, the units in their model represent the collective activity of a column, reducing the network dynamics to a firing-rate approximation. But this simplification ignores the rich temporal dynamics of spiking networks, which are known to affect bump stability. More fundamentally, there is no role for functionally distinct neuron types.

The primary criticism of the Ernst et al.'s bump model is that its input only consists of a luminance channel, and it is not obvious how to replace this channel with ON and OFF rectified channels to account for simple and complex cells. One possibility would be to segregate ON-driven and OFF-driven cells (referred to as simple cells in this paper) into two distinct recurrent networks. Because each network would have its own innate noise profile, bumps would form independently. Consequently, there is no guarantee that ON-driven maps would line up with OFF-driven maps, which would result in conflicting orientation signals when these simple cells converge onto sign-independent (complex) cells.

## 2.2    Simple Cells Solve a Complex Problem

To ensure that both ON-driven and OFF-driven simple cells have the same orientation maps, both ON and OFF bumps must be computed in the same recurrent network so that they are subjected to the same noise profile. We achieve this by building our recurrent network out of cells that are sign-independent; that is both ON and OFF channels drive the network. These cells exhibit complex cell-like behavior (and are referred to as complex cells in this paper) because they are modulated at double the spatial frequency of a sinusoidal grating input. The simple cells subsequently derive their responses from two separate signals: an orientation selective feedback signal from the complex cells indicating the presence of either an ON or an OFF bump, and an ON–OFF selection signal that chooses the appropriate response flavor.

Figure 1 *left* illustrates the formation of bumps (highlighted cells) by a recurrent network with a Mexican hat connectivity profile. Extending the Ernst et al. model, these complex bumps seed simple bumps when driven by a grating. Simple bumps that match the sign of the input survive, whereas out-of-phase bumps are extinguished (faded cells) by push-pull inhibition.

Figure 1 *right* shows the local connections within a microcircuit. An EXC (excitatory) cell receives excitatory input from both ON and OFF channels, and

projects to other EXC (not shown) and INH (inhibitory) cells. The INH cell projects back in a reciprocal configuration to EXC cells. The divergence is indicated in *left*. ON-driven and OFF-driven simple cells receive input in a push-pull configuration (i.e., ON cells are excited by ON inputs and inhibited by OFF inputs, and vise-versa), while additionally receiving input from the EXC–INH recurrent network. In this model, we implement our push-pull circuit using monosynaptic inhibitory connections, despite the fact that geniculate input is strictly excitatory. This simplification, while anatomically incorrect, yields a more efficient implementation that is functionally equivalent.

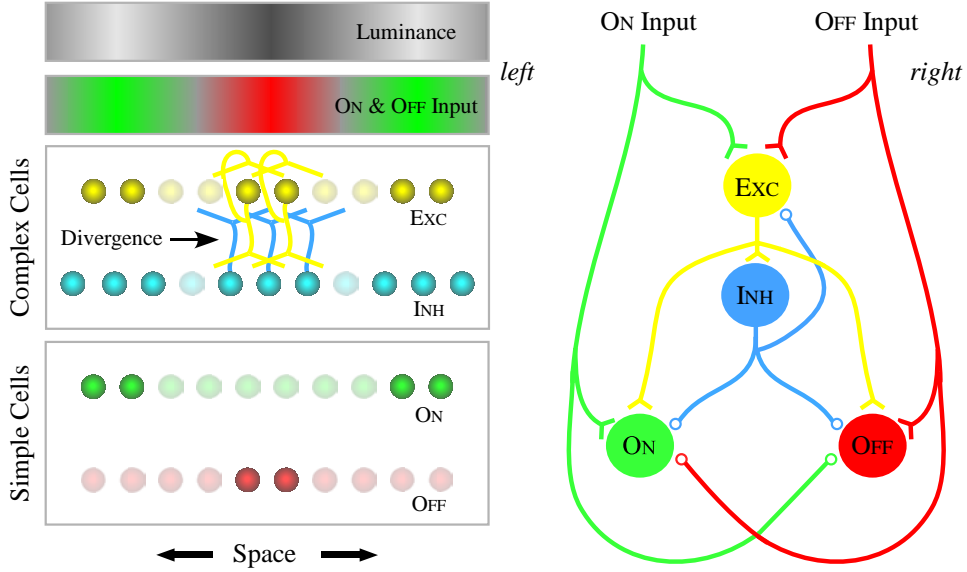

Figure 1: *left*, Complex and simple cell responses to a sinusoidal grating input. Luminance is transformed into ON (green) and OFF (red) pathways by retinal processing. Complex cells form a recurrent network through excitatory and inhibitory projections (yellow and blue lines, respectively), and clusters of activity occur at twice the spatial frequency of the grating. ON input activates ON-driven simple cells (bright green) and suppresses OFF-driven simple cells (faded red), and vise-versa. *right*, The bump model's local microcircuit: circles represent neurons, curved lines represent axon arbors that end in excitatory synapses (v shape) or inhibitory synapses (open circles). For simplicity, inhibitory interneurons were omitted in our push-pull circuit.

## 2.3   Mathematical Description

The neurons in our network follow the equation $C\dot{V} = -\sum_n \delta(t - t_n) + I_{\text{syn}} - I_{\text{KCa}} - I_{\text{leak}}$, where $C$ is membrane capacitance, $\dot{V}$ is the temporal derivative of the membrane voltage, $\delta(\cdot)$ is the Dirac delta function, which resets the membrane at the times $t_n$ when it crosses threshold, $I_{\text{syn}}$ is synaptic current from the network, and $I_{\text{leak}}$ is a constant leak current. Neurons receive synaptic current of the form:

$$I_{\text{syn}}^{\text{ON}} = w^+ I^{\text{ON}} - w^- I^{\text{OFF}} + w^{\text{EE}} I^{\text{EXC}} - w^{\text{EI}} I^{\text{INH}}, \quad I_{\text{syn}}^{\text{EXC}} = w^+(I^{\text{ON}} + I^{\text{OFF}}) + w^{\text{EE}} I^{\text{EXC}} - w^{\text{EI}} I^{\text{INH}} + I_{\text{back}},$$

$$I_{\text{syn}}^{\text{OFF}} = w^+ I^{\text{OFF}} - w^- I^{\text{ON}} + w^{\text{EE}} I^{\text{EXC}} - w^{\text{EI}} I^{\text{INH}}, \quad I_{\text{syn}}^{\text{INH}} = w^{\text{IE}} I^{\text{EXC}}$$

where $w^+$ is the excitatory synaptic strength for ON and OFF input synapses, $w^-$ is the strength of the push-pull inhibition, $w^{EE}$ is the synaptic strength for EXC cell projections to other EXC cells, $w^{EI}$ is the strength of INH cell projections to EXC cells, $w^{IE}$ is the strength of EXC cell projections to INH cells, $I_{back}$ is a constant input current, and $I^{\{ON,OFF,EXC,INH\}}$ account for all impinging synapses from each of the four cell types. These terms are calculated for cell $i$ using an arbor function that consists of a spatial weighting $J(r)$ and a post-synaptic current waveform $\alpha(t)$: $\sum_{k,n} J(i-k) \cdot \alpha(t-t_n^k)$, where $k$ spans all cells of a given type and $n$ indexes their spike times. The spatial weighting function is described by $J(i-k) = \exp(-|i-k|/\sigma)$, with $\sigma$ as the space constant. The current waveform, which is non-zero for $t>0$, convolves a $1/t$ function with a decaying exponential: $\alpha(t) = (t/\tau_c + \alpha_0)^{-1} * \exp(-t/\tau_e)$, where $\tau_c$ is the decay-rate, and $\tau_e$ is the time constant of the exponential. Finally, we model spike-rate adaptation with a calcium-dependent potassium-channel (KCa), which integrates Ca triggered by spikes at times $t_n$ with a gain $K$ and a time constant $\tau_k$, as described by $I_{KCa} = \sum_n K \exp(t_n - t/\tau_k)$.

# 3   Silicon Implementation

We implemented our model in silicon using the TSMC (Taiwan Semiconductor Manufacturing Company) 0.25μm 5-metal layer CMOS process.  The final chip consists of a 2-D core of 48x48 pixels, surrounded by asynchronous digital circuitry that transmits and receives spikes in real-time.  Neurons that reach threshold within the array are encoded as address-events and sent off-chip, and concurrently, incoming address-events are sent to their appropriate synapse locations.  This interface is compatible with other spike-based chips that use address-events [5]. The fabricated bump chip has close to 460,000 transistors packed in 10 mm$^2$ of silicon area for a total of 9,216 neurons.

## 3.1   Circuit Design

Our neural circuit was morphed into hardware using four building blocks.  Figure 2 shows the transistor implementation for synapses, axonal arbors (diffuser), KCa analogs, and neurons.  The circuits are designed to operate in the subthreshold region (except for the spiking mechanism of the neuron).  Noise is not purposely designed into the circuits.  Instead, random variations from the fabrication process introduce significant deviations in I-V curves of theoretically identical MOS transistors.

The function of the synapse circuit is to convert a brief voltage pulse (neuron spike) into a postsynaptic current with biologically realistic temporal dynamics.  Our synapse achieves this by cascading a current-mirror integrator with a log-domain low-pass filter.  The current-mirror integrator has a current impulse response that decays as $1/t$ (with a decay rate set by the voltage $\tau_c$ and an amplitude set by A). This time-extended current pulse is fed into a log-domain low-pass filter (equivalent to a current-domain RC circuit) that imposes a rise-time on the post-synaptic current set by $\tau_e$.  ON and OFF input synapses receive presynaptic spikes from the off-chip link, whereas EXC and INH synapses receive presynaptic spikes from local on-chip neurons.

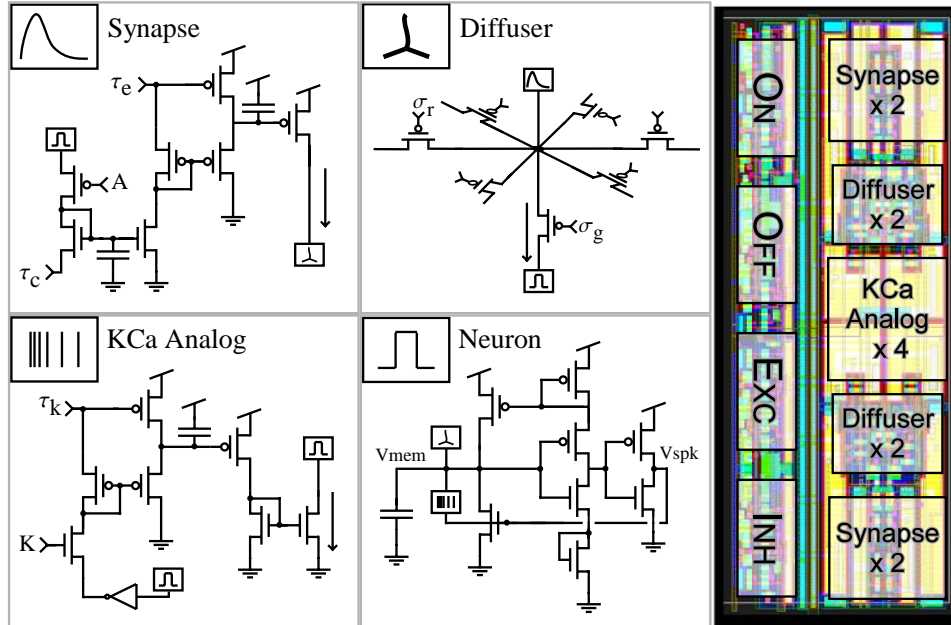

Figure 2: Transistor implementations are shown for a synapse, diffuser, KCa analog, and neuron (simplified), with circuit insignias in the top-left of each box. The circuits they interact with are indicated (e.g. the neuron receives synaptic current from the diffuser as well as adaptation current from the KCa analog; the neuron in turn drives the KCa analog). The far right shows layout for one pixel of the bump chip (vertical dimension is 83μm, horizontal is 30 μm).

The diffuser circuit models axonal arbors that project to a local region of space with an exponential weighting. Analogous to resistive divider networks, diffusers [6] efficiently distribute synaptic currents to multiple targets. We use four diffusers to implement axonal projections for: the ON pathway, which excites ON and EXC cells and inhibits OFF cells; the OFF pathway, which excites OFF and EXC cells and inhibits ON cells; the EXC cells, which excite all cell types; and the INH cells, which inhibits EXC, ON, and OFF cells. Each diffuser node connects to its six neighbors through transistors that have a pseudo-conductance set by $\sigma_r$, and to its target site through a pseudo-conductance set by $\sigma_g$; the space-constant of the exponential synaptic decay is set by $\sigma_r$ and $\sigma_g$'s relative levels.

The neuron circuit integrates diffuser currents on its membrane capacitance. Diffusers either directly inject current (excitatory), or siphon off current (inhibitory) through a current-mirror. Spikes are generated by an inverter with positive feedback (modified from [7]), and the membrane is subsequently reset by the spike signal. We model a calcium concentration in the cell with a KCa analog. K controls the amount of calcium that enters the cell per spike; the concentration decays exponentially with a time constant set by $\tau_k$. Elevated charge levels activate a KCa-like current that throttles the spike-rate of the neuron.

## 3.2  Experimental Setup

Our setup uses either a silicon retina [8] or a National Instruments DIO (digital input–output) card as input to the bump chip. This allows us to test our V1 model with real-time visual stimuli, similar to the experimental paradigm of

electrophysiologists. More specifically, the setup uses an address-event link [5] to establish virtual point-to-point connectivity between ON or OFF ganglion cells from the retina chip (or DIO card) with ON or OFF synapses on the bump chip. Both the input activity and the output activity of the bump chip is displayed in real-time using receiver chips, which integrate incoming spikes and displays their rates as pixel intensities on a monitor. A logic analyzer is used to capture spike output from the bump chip so it can be further analyzed.

We investigated responses of the bump chip to gratings moving in sixteen different directions, both qualitatively and quantitatively. For the qualitative aspect, we created a PO map by taking each cell's average activity for each stimulus direction and computing the vector sum. To obtain a quantitative measure, we looked at the normalized vector magnitude (NVM), which reveals the sharpness of a cell's tuning. The NVM is calculated by dividing the vector sum by the magnitude sum for each cell. The NVM is 0 if a cell responds equally to all orientations, and 1 if a cell's orientation selectivity is perfect such that it only responds at a single orientation.

# 4   Results

We presented sixteen moving gratings to the network, with directions ranging from 0 to 360 degrees. The spatial frequency of the grating is tuned to match the size of the average bump, and the temporal frequency is 1 Hz. Figure 3a shows a resulting PO map for directions from 180 to 360 degrees, looking at the inhibitory cell population (the data looks similar for other cell types). Black contours represent stable bump regions, or equivalently, the regions that exceed a prescribed threshold (90 spikes) for all directions. The PO map from the bump chip reveals structure that resembles data from real cortex. Nearby cells tend to prefer similar orientations except at fractures. There are even regions that are similar to pinwheels (delimited by a white rectangle).

A PO is a useful tool to describe a network's selectivity, but it only paints part of the picture. So we have additionally computed a NVM map and a NVM histogram, shown in Figure 3b and 3c respectively. The NVM map shows that cells with sharp selectivity tend to cluster, particularly around the edge of the bumps. The histogram also reveals that the distribution of cell selectivity across the network varies considerably, skewed towards broadly tuned cells.

We also looked at spike rasters from different cell-types to gain insight into their phase relationship with the stimulus. In particular, we present recordings for the site indicated by the arrow (see Figure 3a) for gratings moving in eight directions ranging from 0 to 360 degrees in 45-degree increments (this location was chosen because it is in the vicinity of a pinwheel, is reasonably selective, and shows considerable modulation in its firing rate). Figure 4 shows the luminance of the stimulus (bottom sinusoids), ON- (cyan) and OFF-input (magenta) spike trains, and the resulting spike trains from EXC (yellow), INH (blue), ON- (green), and OFF-driven (red) cell types for each of the eight directions. The center polar plot summarizes the orientation selectivity for each cell-type by showing the normalized number of spikes for each stimulus. Data is shown for one period.

Even though all cells-types are selective for the same orientation (regardless of grating direction), complex cell responses tend to be phase-insensitive while the simple cell responses are modulated at the fundamental frequency. It is worth noting that the simple cells have sharper orientation selectivity compared to the complex cells. This trend is characteristic of our data.

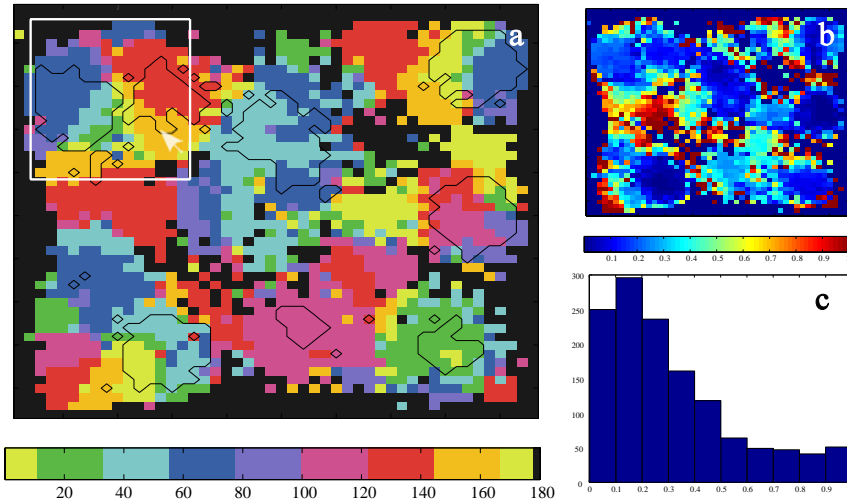

Figure 3: (a) PO map for the inhibitory cell population stimulated with eight different directions from 180 to 360 degrees (black represents no activity, contours delineate regions that exceed 90 spikes for all stimuli). Normalized vector magnitude (NVM) data is presented as (b) a map and (c) a histogram.

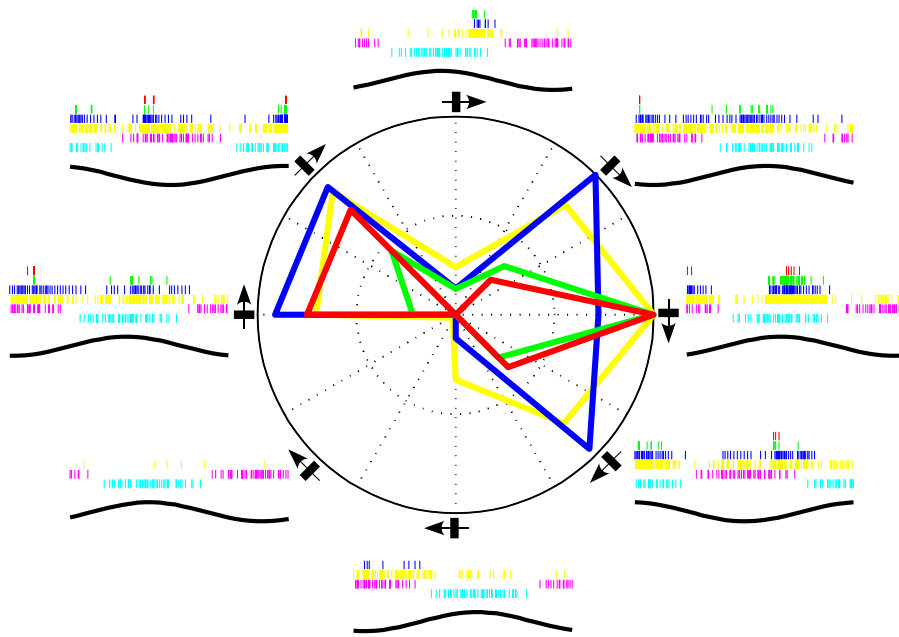

Figure 4: Spike rasters and polar plot for 8 directions ranging from 0 to 360 degrees. Each set of spike rasters represent from bottom to top, ON- (cyan) and OFF-input (magenta), INH (yellow), EXC (blue), and ON- (green) and OFF-driven (red). The stimulus period is 1 sec.

# 5 Discussion

We have implemented a large-scale network of spiking neurons in a silicon chip that is based on layer 4 of the visual cortex. The initial testing of the network reveals a PO map, inherited from innate chip heterogeneities, resembling cortical maps. Our microcircuit proposes a novel function for complex-like cells; that is they create a sign-independent orientation selective signal, which through a push-pull circuit creates sharply tuned simple cells with the same orientation preference.

Recently, Ringach et al. surveyed orientation selectivity in the macaque [9]. They observed that, in a population of V1 neurons (N=308) the distribution of orientation selectivity is quite broad, having a median NVM of 0.39. We have measured median NVM's ranging from 0.25 to 0.32. Additionally, Ringach et al. found a negative correlation between spontaneous firing rate and NVM. This is consistent with our model because cells closer to the center of the bump have higher firing rates and broader tuning.

While the results from the bump chip are promising, our maps are less consistent and noisier than the maps Ernst et al. have reported. We believe this is because our network is tuned to operate in a fluid state where bumps come on, travel a short distance and disappear (motivated by cortical imaging studies). But excessive fluidity can cause non-dominant bumps to briefly appear and adversely shift the PO maps. We are currently investigating the role of lateral connections between bumps as a means to suppress these spontaneous shifts.

The neural mechanisms that underlie the orientation selectivity of V1 neurons are still highly debated. This may be because neuron responses are not only shaped by feedforward inputs, but are also influenced at the network level. If modeling is going to be a useful guide for electrophysiologists, we must model at the network level while retaining cell level detail. Our results demonstrate that a spike-based neuromorphic system is well suited to model layer 4 of the visual cortex. The same approach may be used to build large-scale models of other cortical regions.

# References

1. Hubel, D. and T. Wiesel, *Receptive firelds, binocular interaction and functional architecture in the cat's visual cortex.* J. Physiol, 1962. **160**: p. 106-154.
2. Blasdel, G.G., *Orientation selectivity, preference, and continuity in monkey striate cortex.* J Neurosci, 1992. **12**(8): p. 3139-61.
3. Crair, M.C., D.C. Gillespie, and M.P. Stryker, *The role of visual experience in the development of columns in cat visual cortex.* Science, 1998. **279**(5350): p. 566-70.
4. Ernst, U.A., et al., *Intracortical origin of visual maps.* Nat Neurosci, 2001. **4**(4): p. 431-6.
5. Boahen, K., *Point-to-Point Connectivity. IEEE Transactions on Circuits & Systems II*, 2000. **vol 47 no 5**: p. 416-434.
6. Boahen, K. and Andreou. *A contrast sensitive silicon retina with reciprocal synapses.* in *NIPS91*. 1992: IEEE.
7. Culurciello, E., R. Etienne-Cummings, and K. Boahen, *A Biomorphic Digital Image Sensor.* IEEE Journal of Solid State Circuits, 2003. **vol 38 no 2**: p. 281-294.
8. Zaghloul, K., *A silicon implementation of a novel model for retinal processing*, in *Neuroscience*. 2002, UPENN: Philadelphia.
9. Ringach, D.L., R.M. Shapley, and M.J. Hawken, *Orientation selectivity in macaque V1: diversity and laminar dependence.* J Neurosci, 2002. **22**(13): p. 5639-51.
